# Tempo Tracking and Rhythm Quantization by Sequential Monte Carlo

**Ali Taylan Cemgil and Bert Kappen**
SNN, University of Nijmegen
NL 6525 EZ Nijmegen
The Netherlands
{cemgil,bert}@mbfys.kun.nl

## Abstract

We present a probabilistic generative model for timing deviations in expressive music performance. The structure of the proposed model is equivalent to a switching state space model. We formulate two well known music recognition problems, namely tempo tracking and automatic transcription (rhythm quantization) as filtering and maximum a posteriori (MAP) state estimation tasks. The inferences are carried out using sequential Monte Carlo integration (particle filtering) techniques. For this purpose, we have derived a novel Viterbi algorithm for Rao-Blackwellized particle filters, where a subset of the hidden variables is integrated out. The resulting model is suitable for realtime tempo tracking and transcription and hence useful in a number of music applications such as adaptive automatic accompaniment and score typesetting.

## 1 Introduction

Automatic music transcription refers to extraction of a high level description from musical performance, for example in form of a music notation. Music notation can be viewed as a list of the pitch levels and corresponding timestamps.

Ideally, one would like to recover a score directly from sound. Such a representation of the surface structure of music would be very useful in music information retrieval (Music-IR) and content description of musical material in large audio databases. However, when operating on sampled audio data from polyphonic acoustical signals, extraction of a score-like description is a very challenging auditory scene analysis task [13].

In this paper, we focus on a subproblem in music-ir, where we assume that exact timing information of notes is available, for example as a stream of MIDI[1] events

from a digital keyboard.

A model for tempo tracking and transcription is useful in a broad spectrum of applications. One example is automatic score typesetting, the musical analog of word processing. Almost all score typesetting applications provide a means of automatic generation of a conventional music notation from MIDI data.

In conventional music notation, onset time of each note is implicitly represented by the cumulative sum of durations of previous notes. Durations are encoded by simple rational numbers (e.q. quarter note, eight note), consequently all events in music are placed on a discrete grid. So the basic task in MIDI transcription is to associate discrete grid locations with onsets, i.e. quantization.

However, unless the music is performed with mechanical precision, identification of the correct association becomes difficult. Consequently resulting scores have often very poor quality. This is due to the fact that musicians introduce intentional (and unintentional) deviations from a mechanical prescription. For example timing of events can be deliberately delayed or pushed. Moreover, the tempo can fluctuate by slowing down or accelerating. In fact, such deviations are natural aspects of expressive performance; in the absence of these, music tends to sound rather dull.

Robust and fast quantization and tempo tracking is also an important requirement in interactive performance systems. These are emerging applications that "listen" to the performance for generating an accompaniment or improvisation in real time [10, 12]. At last, such models are also useful in musicology for systematic study and characterization of expressive timing by principled analysis of existing performance data.

## 2 Model

Consider the following generative model for timing deviations in music

$$
\begin{aligned}
c_k &= c_{k-1} + \gamma_{k-1} & (1) \\
\omega_k &= \omega_{k-1} + \zeta_k & (2) \\
\tau_k &= \tau_{k-1} + 2^{\omega_k}(c_k - c_{k-1}) & (3) \\
y_k &= \tau_k + \epsilon_k & (4)
\end{aligned}
$$

In Eq. 1, $c_k$ denotes the grid location of $k$'th onset in a score. The interval between two consecutive onsets in the score is denoted by $\gamma_{k-1}$. For example consider the notation ♩ ♫ which encodes $\gamma_{1:3} = [1 \quad 0.5 \quad 0.5]$, hence $c_{1:4} = [0 \quad 1 \quad 1.5 \quad 2]$. We assign a prior of form $p(c_k) \propto \exp(-d(c_k))$ where $d(c_k)$ is the number of significant digits in the binary expansion of the fraction of $c_k$ [1]. One can check that such a prior prefers simpler notations, e.g. $p(\; \text{♫♫} \;) < p(\; ♩ \; ♫ \;)$. We note that $c_k$ are drawn from an infinite (but discrete) set and are increasing in $k$, i.e $c_k \geq c_{k-1}$. To allow for different time signatures and alternative rhythmic subdivisions, one can introduce additional hidden variables [1], but this is not addressed in this paper.

Eq. 2 defines a prior over possible tempo deviations. We denote the logarithm of the period (inverse tempo) by $\omega$. For example if the tempo is 60 beats per minute (bpm), $\omega = \log 1\text{sec} = 0$. Since tempo appears as a scale variable in mapping grid locations on a score to the actual performance time, we have chosen to represent it in the logarithmic scale (eventually a gamma distribution can also be used). This representation is both perceptually plausible and mathematically convenient since a symmetric noise model on $\omega$ assigns equal probabilities to equal *relative* changes in tempo. We take $\zeta_k$ to be a Gaussian random variable with $\mathcal{N}(0, \lambda^2 \gamma_k Q)$. Depending

upon the interval between consecutive onsets, the model scales the noise covariance; longer jumps in the score allow for more freedom in fluctuating the tempo. Given the $\omega$ sequence, Eq. 3 defines a model of noiseless onsets with variable tempo. We will denote the pair of hidden continuous variables by $z_k = (\tau_k, \omega_k)$.

Eq. 4 defines the observation model. Here $y_k$ is the observed onset time of the $k$'th onset in the performance. The noise term $\epsilon_k$ models small scale expressive deviations in timing of individual notes and has a Gaussian distribution parameterized by $\mathcal{N}(\mu(\gamma_{k-1}), \Sigma(\gamma_{k-1}))$. Such a parameterization is useful for appropriate quantization of phrases (short sequences of notes) that are shifted or delayed as a whole [1].

In reality, a random walk model for tempo such as in Eq. 2 is not very realistic. Tempo deviations are usually more smooth. In the dynamical model framework such smooth deviations can be allowed by increasing the dimensionality of $\omega$ by include higher order "inertia" variables [2]. In this case we simply rewrite Eq. 2 as

$$\omega_k \quad = \quad A\omega_{k-1} + \zeta_k$$

and take a diagonal $Q$. Accordingly, the observation model (Eq. 4) changed such that $\omega_k$ is replaced by $C\omega_k$ where $C = [1\ 0\dots0]$.

The graphical model is shown in Figure 1. The model is similar to a switching state space model, that has been recently applied in the context of music transcription [11]. The differences are in parameterization and more importantly in the inference method.

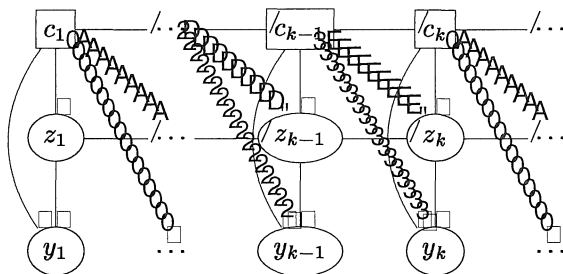

Figure 1: Graphical Model. The pair of continuous hidden variables $(\tau_k, \omega_k)$ is denoted by $z_k$. Both $c$ and $z$ are hidden; only the onsets $y$ are observed.

We define tempo tracking as a filtering problem

$$z_k^* \quad = \quad \underset{z_k}{\operatorname{argmax}} \sum_{c_k} p(c_k, z_k | y_{1:k}) \tag{5}$$

and rhythm transcription as a MAP state estimation problem

$$c_{1:K}^* \quad = \quad \underset{c_{1:K}}{\operatorname{argmax}} \, p(c_{1:K} | y_{1:K}) \tag{6}$$

$$p(c_{1:K} | y_{1:K}) \quad = \quad \int dz_{1:K} p(c_{1:K}, z_{1:K} | y_{1:K}) \tag{7}$$

The exact computation of the quantities in Eq. 6 and Eq. 5 is intractable due to the explosion in the number of mixture components required to represent the exact posterior at each step $k$. Consequently we will use Monte Carlo approximation techniques.

# 3   Sequential Monte Carlo Sampling

Sequential Monte Carlo sampling (a.k.a. particle filtering) is an integration method especially powerful for inference in dynamical systems. See [4] for a detailed review of state of the art. At each step $k$, the exact marginal posterior over hidden states $x_k$ is approximated by an empirical distribution of form

$$p(x_k|y_{1:k}) \quad \approx \quad \sum_{i=1}^{N} w_k^{(i)} \delta(x_k - x_k^{(i)}) \tag{8}$$

where $x_k^{(i)}$ are a set of points obtained by sampling from a proposal distribution and $w_k^{(i)}$ are associated importance weights such that $\sum_{i=1}^{N} w_k^{(i)} = 1$. Particles at step $k$ are evolved to $k+1$ by sequential importance sampling and resampling methods [6]. Once a set of discrete sample points is obtained during the forward phase by sampling, particle approximations to quantities such as the smoothed marginal posterior $p(x_k|y_{1:K})$ or the maximum a posteriori state sequence (Viterbi path) $x_{1:K}^*$ can be obtained efficiently. Due to the discrete nature of the approximate representation, resulting algorithms are closely related to standard smoothing and Viterbi algorithms in Hidden Markov models [9, 7, 6].

Unfortunately, if the hidden state space is of high dimensionality, sampling can be inefficient. Hence increasingly many particles are needed to accurately represent the posterior. Consequently, the estimation of "off-line" quantities such as $p(x_k|y_{1:K})$ and $x_{1:K}^*$ becomes very costly since one has to store all past trajectories.

For some models, including the one proposed here, one can identify substructures where integrations, conditioned on certain nodes can be computed analytically [5]. Conditioned on $c_{1:k}$, the model reduces to the (extended) [2] Kalman filter. In this case the joint marginal posterior is represented as a mixture

$$p(c_k, z_k|y_{1:k}) \quad \approx \quad \sum_{i=1}^{N} w_k^{(i)} p(z_k|c_k^{(i)}, y_{1:k}) \delta(c_k - c_k^{(i)}) \tag{9}$$

The particular case of Gaussian $p(z_k|c_k^{(i)}, y_{1:k})$ is extensively used in diverse applications [8] and reported to give superior results when compared to standard particle filtering [3, 6].

## 3.1   Particle Filtering

We assume that we have obtained a set of particles from filtered posterior $p(c_k|y_{1:k})$. Due to lack of space we do not give the details of the particle filtering algorithm but refer the reader to [6]. One important point to note is that we have to use the optimal proposal distribution given as

$$\hat{p}(c_k|c_{k-1}^{(i)}, y_{1:k}) \propto \int dz_{k-1:k}\ p(y_k|z_k, c_k, c_{k-1}^{(i)})$$
$$p(z_k, c_k|z_{k-1}, c_{k-1}^{(i)}) p(z_{k-1}|c_{k-1}^{(i)}, y_{1:k-1}) \tag{10}$$

Since the state-space of $c_k$ is effectively infinite, this step is crucial for efficiency. Evaluation of the proposal distribution amounts to looking forward and selecting a set of high probability candidate grid locations for quantization. Once $c_k^{(i)}$ are obtained we can use standard Kalman filtering algorithms to update the Gaussian potentials $p(z_k|c_k^{(i)}, y_{1:k})$. Thus tempo tracking problem as stated in Eq. 5 is readily solved.

## 3.2 Modified Viterbi algorithm

The quantization problem in Eq. 6 can only be solved approximately. Since $z$ is integrated over, in general all $c_k$ become coupled and the Markov property is lost, i.e. $p(c_{1:K}|y_{1:K})$ is in general not a chain. One possible approximation, that we adapt also here, is to assume smoothed estimates are not much different from filtered estimates [8] i.e.

$$p(c_k, z_k|c_{k-1}, z_{k-1}, y_{1:K}) \approx p(c_k, z_k|c_{k-1}, z_{k-1}, y_{1:k}) \qquad (11)$$

and to write

$$p(c_{1:K}|y_{1:K}) \approx \int dz_{1:K} p(c_1 z_1|y_1) \prod_{k=2}^{K} p(c_k, z_k|c_{k-1}, z_{k-1}, y_{1:k})$$

$$\propto \int dz_{1:K} p(y_1|z_1, c_1) p(z_1, c_1) \prod_{k=2}^{K} p(y_k|z_k, c_k, c_{k-1}) p(z_k, c_k, |z_{k-1}, c_{k-1})$$

If we plug in the mixture approximation in Eq. 9 and take the $\arg\max \log$ on both sides we obtain a sum that can be stepwise optimized using the Viterbi algorithm [9].

The standard Viterbi algorithm for particle filters [7] defines a transition matrix $T_{k-1} = f(c_k^{(j)}|c_{k-1}^{(i)})$ between each pair of particles at consecutive time slices. Here, $f$ is a state transition distribution that can be evaluated pointwise and $T_{k-1}$ can be computed on the fly by evaluating $f$ at $(c_k^{(j)}, c_{k-1}^{(i)})$. In contrast, the modified Viterbi algorithm replaces the pointwise evaluation by an expectation under $p(z_k, z_{k-1}|c_k^{(j)}, c_{k-1}^{(i)}, y_{1:k})$ where the transition matrix is defined as $T_{k-1}(j,i) = \hat{p}(c_k^{(j)}|c_{k-1}^{(i)}, y_{1:k})$. In this case, each entry of $T$ is computed by one step Kalman likelihood evaluation.

1. Initialization. For $i = 1 : N$
$$\delta_1(i) = \log p(c_1^{(i)}) + \log \hat{p}(y_1|c_1^{(i)})$$

2. Recursion. For $j = 1 : N$, $k = 2 : K$
$$\begin{aligned} T_{k-1}(j,i) &= \log \hat{p}(c_k^{(j)}|c_{k-1}^{(i)}, y_{1:k}) \quad \text{(See Eq. 10)} \\ \delta_k(j) &= \max_i \{\delta_{k-1}(i) + T_{k-1}(j,i)\} \\ \psi_k(j) &= \arg\max_i \{\delta_{k-1}(i) + T_{k-1}(j,i)\} \end{aligned}$$

3. Termination.
$$\begin{aligned} r_K &= \arg\max_i \delta_K(i) \\ c_K^* &= c_K^{(r_K)} \end{aligned}$$

4. Backtracking. For $k = K - 1 : -1 : 1$
$$\begin{aligned} r_k &= \psi_{k+1}(r_{k+1}) \\ c_k^* &= c_k^{(r_k)} \end{aligned}$$

Since the tempo trajectories can be integrated out online, we need to only store the links $\psi_k$ and quantization locations $c_k^{(i)}$. Consequently, the random walk tempo prior can be replaced by a richer model as in Eq. 5, virtually without additional computational or storage cost. An outline of the algorithm is shown in Figure 2. Of course, the efficiency and accuracy of our approach depends heavily onto the assumption in Eq. 11, that the $T$ matrix based on filtered estimates is accurate.

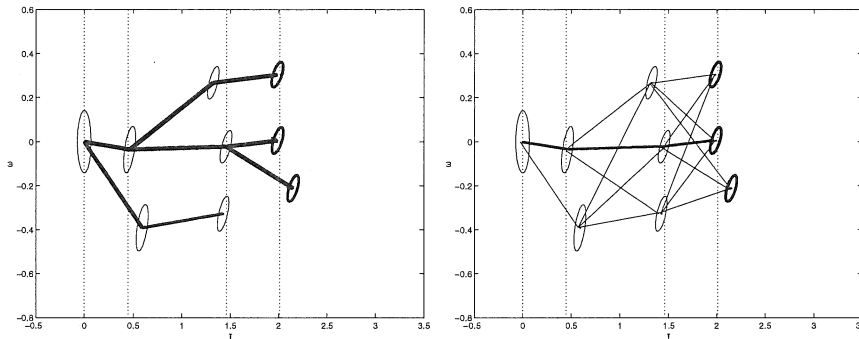

Figure 2: Outline of the algorithm. Left: forward filtering phase. The ellipses correspond to the conditionals $p(z_k|c_k^{(n)}, y_{1:k})$. Vertical dotted lines denote the observations $y_k$. At each step $k$, particles with low likelihood are discarded. Surviving particles are linked to their parents. Right: The transition matrix $T_{k-1}$ between each generation (forall pairs of $c_k^{(j)}, c_{k-1}^{(i)}$) is computed by standard Kalman filter likelihood equations. Note that $T_{k-1}$ can be discarded once the forward messages $\delta_k$ are computed and only the backward links $\psi_{1:K}$ and corresponding $c_k$ need to be stored. When all onsets $y_{1:K}$ are observed, the MAP sequence $c_{1:K}^*$ is computed by backtracking.

## 4  Simulation Results

We have tested tempo tracking and quantization performance of the model on two different examples. The first example is a repeating "son-clave" pattern ‖: ♩ ♪ ♪ ♩ |♩. ♩. ♩ :‖ ($c = [1\ \ 2\ \ 4\ \ 5.5\ \ 7\dots]$) with fluctuating tempo [3]. Such syncopated rhythms are usually hard to transcribe and make it difficult to track the tempo even for experienced human listeners. Moreover, since onsets are absent at prominent beat locations, standard beat tracking algorithms usually loose track.

We observe that for various realistic tempo fluctuations and observation noise level, the particle filter is able to identify the correct tempo trajectory and the corresponding quantization (Figure 3, above).

The second example is a piano arrangement of the Beatles song (Yesterday) performed by a professional classical pianist on a MIDI grand piano. This is a polyphonic piece, i.e. the arrangement contains chords and events occurring at the same time. We model polyphony by allowing $c_k - c_{k-1} = 0$. In this case, since the original arrangement is known, we estimate the true tempo trajectory by Kalman filtering after clamping $c_{1:K}$. As shown in Figure 3, the particle filter estimate and the true tempo trajectory are almost identical.

## 5  Discussion and Conclusion

There are several advantages offered by particle filtering approach. The algorithm is suitable for real time implementation. Since the implementation is easy, this provides an important flexibility in the models one can employ. Although we have not

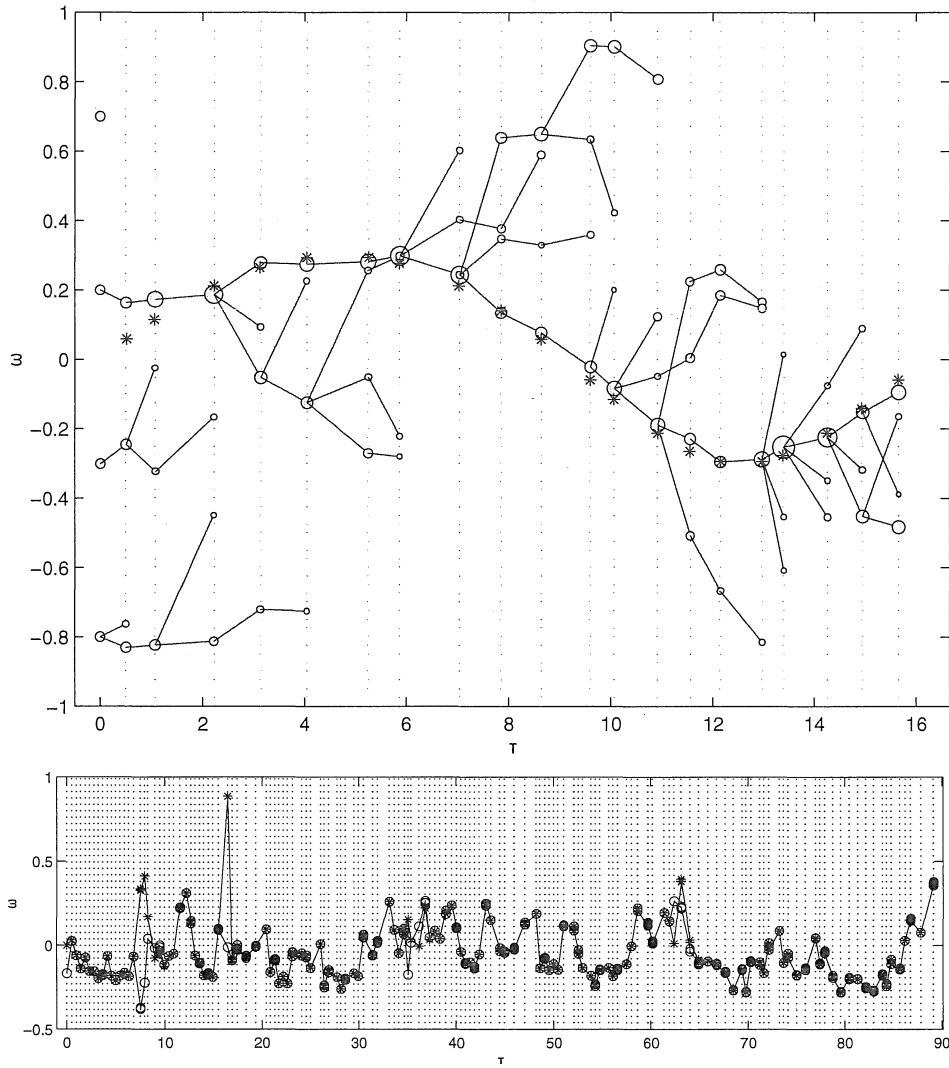

Figure 3: Above: Tempo tracking results for the clave pattern with 4 particles. Each circle denotes the mean $(\tau_k^{(n)}, \omega_k^{(n)})$. The diameter of each particle is proportional to the normalized importance weight at each generation. '*' denote the true $(\tau, \omega)$ pairs. Below: Tracking results for "Yesterday". '*' denote the mean of the filtered $z_{1:K}$ after clamping to true $c_{1:K}$. Small circles denote the mean $z_{1:K}$ corresponding to the estimated MAP trajectory $c_{1:K}^*$ using 10 particles.

addressed issues such as learning and online adaptation in this paper, parameters of the model can also treated as hidden variables and can be eventually integrated out similar to the tempo trajectories.

Especially in real time music applications fine tuning and careful allocation of computational resources is of primary importance. Particle filtering is suitable since one can simply reduce the number of particles when computational resources become overloaded.

Motivated by the advantages of the particle filtering approach, we are currently working on a real time implementation of the particle filter based tempo tracker for eventual automatic accompaniment generation such as an adaptive drum machine. Consequently, the music is quantized such that it can be typeset in a notation program.

## Acknowledgements

This research is supported by the Technology Foundation STW, applied science division of NWO and the technology programme of the Dutch Ministry of Economic Affairs.

## Footnotes

[1] Musical Instruments Digital Interface. A standard communication protocol especially designed for digital instruments such as keyboards. Each time a key is pressed, a MIDI keyboard generates a short message containing pitch and key velocity. A computer can tag each received message by a timestamp for real-time processing and/or "recording" into a file.

[2]We linearize the nonlinear observation model $2^{\omega_k}(c_k - c_{k-1})$ around the expectation $\langle \omega_k \rangle$.

[3]We modulate the tempo deterministically according to $\omega_k = 0.3\sin(2\pi c_k/32)$. The observation noise variance is $R = 0.0005$.

## References

[1] A. T. Cemgil, P. Desain, and H. Kappen. Rhythm quantization for transcription. *Computer Music Journal*, 24:2:60–76, 2000.

[2] A. T. Cemgil, H. Kappen, P. Desain, and H. Honing. On tempo tracking: Tempogram representation and kalman filtering. *Journal of New Music Research*, Accepted for Publication.

[3] R. Chen and J. S. Liu. Mixture kalman filters. *J. R. Statist. Soc.*, 10, 2000.

[4] A. Douchet, N. de Freitas, and N. J. Gordon, editors. *Sequential Monte Carlo Methods in Practice*. Springer-Verlag, New York, 2000.

[5] A. Douchet, N. de Freitas, K. Murphy, and S. Russell. Rao-blackwellised particle filtering for dynamic bayesian networks. In *Uncertainty in Artificial Intelligence*, 2000.

[6] A. Douchet, S. Godsill, and C. Andrieu. On sequential monte carlo sampling methods for bayesian filtering. *Statistics and Computing*, 10(3):197–208, 2000.

[7] S. Godsill, A. Douchet, and M. West. Maximum a posteriori sequence estimation using monte carlo particle filters. *Annals of the Institute of Statistical Mathematics.*, 2000.

[8] K. P. Murphy. Switching kalman filters. Technical report, Dept. of Computer Science, University of California, Berkeley, 1998.

[9] L. R. Rabiner. A tutorial in hidden markov models and selected applications in speech recognation. *Proc. of the IEEE*, 77(2):257–286, 1989.

[10] C. Raphael. A probabilistic expert system for automatic musical accompaniment. *Journal of Computational and Graphical Statistics*, Accepted for Publication, 1999.

[11] C. Raphael. A mixed graphical model for rhythmic parsing. In *to appear in Proc. of Uncertainty in AI*, 2001.

[12] B. Thom. Unsupervised learning and interactive jazz/blues improvisation. In *Proceedings of the AAAI2000*. AAAI Press, 2000.

[13] Barry L. Vercoe, William G. Gardner, and Eric D. Scheirer. Structured audio: Creation, transmission, and rendering of parametric sound representations. *Proc. IEEE*, 86:5:922–940, May 1998.
